# A Primal-Dual Message-Passing Algorithm for Approximated Large Scale Structured Prediction

**Tamir Hazan**
TTI Chicago
hazan@ttic.edu

**Raquel Urtasun**
TTI Chicago
rurtasun@ttic.edu

## Abstract

In this paper we propose an approximated structured prediction framework for large scale graphical models and derive message-passing algorithms for learning their parameters efficiently. We first relate CRFs and structured SVMs and show that in CRFs a variant of the log-partition function, known as the soft-max, smoothly approximates the hinge loss function of structured SVMs. We then propose an intuitive approximation for the structured prediction problem, using duality, based on a local entropy approximation and derive an efficient message-passing algorithm that is guaranteed to converge. Unlike existing approaches, this allows us to learn efficiently graphical models with cycles and very large number of parameters.

## 1 Introduction

Unlike standard supervised learning problems which involve simple scalar outputs, structured prediction deals with structured outputs such as sequences, grids, or more general graphs. Ideally, one would want to make joint predictions on the structured labels instead of simply predicting each element independently, as this additionally accounts for the statistical correlations between label elements, as well as between training examples and their labels. These properties make structured prediction appealing for a wide range of applications such as image segmentation, image denoising, sequence labeling and natural language parsing.

Several structured prediction models have been recently proposed, including log-likelihood models such as conditional random fields (CRFs, [10]), and structured support vector machines (structured SVMs) such as maximum-margin Markov networks (M3Ns [21]). For CRFs, learning is done by minimizing a convex function composed of a negative log-likelihood loss and a regularization term. Learning structured SVMs is done by minimizing the convex regularized structured hinge loss.

Despite the convexity of the objective functions, finding the optimal parameters of these models can be computationally expensive since it involves exponentially many labels. When the label structure corresponds to a tree, learning can be done efficiently by using belief propagation as a subroutine; The sum-product algorithm is typically used in CRFs and the max-product algorithm in structured SVMs. In general, when the label structure corresponds to a general graph, one cannot compute the objective nor the gradient exactly, except for some special cases in structured SVMs, such as matching and sub-modular functions [22]. Therefore, one usually resorts to approximate inference algorithms, cf. [2] for structured SVMs and [20, 12] for CRFs. However, the approximate inference algorithms are computationally too expensive to be used as a subroutine of the learning algorithm, therefore they cannot be applied efficiently for large scale structured prediction problems. Also, it is not clear how to define a stopping criteria for these approaches as the objective does not monotonically decrease since the objective and the gradient are both approximated. This might result in poor approximations.

In this paper we propose an approximated structured prediction framework for large scale graphical models and derive message-passing algorithms for learning their parameters efficiently. We relate CRFs and structured SVMs, and show that in CRFs a variant of the log-partition function, known as

soft-max, smoothly approximates the hinge loss function of structured SVMs. We then propose an intuitive approximation for the structured prediction problem, using duality, based on a local entropy approximation and derive an efficient message-passing algorithm that is guaranteed to converge. Unlike existing approaches, this allows us to learn efficiently graphical models with cycles and very large number of parameters. We demonstrate the effectiveness of our approach in an image denoising task. This task was previously solved by sharing parameters across cliques. In contrast, our algorithm is able to efficiently learn large number of parameters resulting in orders of magnitude better prediction.

In the remaining of the paper, we first relate CRFs and structured SVMs in Section 3, show our approximate prediction framework in Section 4, derive a message-passing algorithm to solve the approximated problem efficiently in Section 5, and show our experimental evaluation.

## 2  Regularized Structured Loss Minimization

Consider a supervised learning setting with objects $x \in X$ and labels $y \in \mathcal{Y}$. In structured prediction the labels may be sequences, trees, grids, or other high-dimensional objects with internal structure. Consider a function $\Phi : \mathcal{X} \times \mathcal{Y} \to \mathbb{R}^d$ that maps $(x, y)$ pairs to feature vectors. Our goal is to construct a linear prediction rule

$$y_{\boldsymbol{\theta}}(x) = \operatorname*{argmax}_{y \in \mathcal{Y}} \boldsymbol{\theta}^\top \Phi(x, y)$$

with parameters $\boldsymbol{\theta} \in \mathbb{R}^d$, such that $y_{\boldsymbol{\theta}}(x)$ is a good approximation to the true label of $x$. Intuitively one would like to minimize the loss $\ell(y, y_{\boldsymbol{\theta}})$ incurred by using $\boldsymbol{\theta}$ to predict the label of $x$, given that the true label is $y$. However, since the prediction is norm-insensitive this method can lead to over fitting. Therefore the parameters $\boldsymbol{\theta}$ are typically learned by minimizing the norm-dependent loss

$$\sum_{(x,y) \in \mathcal{S}} \bar{\ell}(\boldsymbol{\theta}, x, y) + \frac{C}{p} \|\boldsymbol{\theta}\|_p^p, \tag{1}$$

defined over a training set $\mathcal{S}$. The function $\bar{\ell}$ is a surrogate loss of the true loss $\ell(y, \hat{y})$. In this paper we focus on structured SVMs and CRFs which are the most common structured prediction models. The first definition of structured SVMs used the structured hinge loss [21]

$$\bar{\ell}_{hinge}(\boldsymbol{\theta}, x, y) = \max_{\hat{y} \in \mathcal{Y}} \left\{ \ell(y, \hat{y}) + \boldsymbol{\theta}^\top \Phi(x, \hat{y}) - \boldsymbol{\theta}^\top \Phi(x, y) \right\}$$

The structured hinge loss upper bounds the true loss function, and corresponds to a maximum-margin approach that explicitly penalizes training examples $(x, y)$ for which $\boldsymbol{\theta}^\top \Phi(x, y) < \ell(y, y_{\boldsymbol{\theta}}(x)) + \boldsymbol{\theta}^\top \Phi(x, y_{\boldsymbol{\theta}}(x))$.

The second loss function that we consider is based on log-linear models, and is commonly used in CRFs [10]. Let the conditional distribution be

$$p(\hat{y}|x, y; \boldsymbol{\theta}) = \frac{1}{Z(x, y)} \exp\left(\ell(y, \hat{y}) + \boldsymbol{\theta}^\top \Phi(x, \hat{y})\right), \qquad Z(x, y) = \sum_{\hat{y} \in \mathcal{Y}} \exp\left(\ell(y, \hat{y}) + \boldsymbol{\theta}^\top \Phi(x, \hat{y})\right)$$

where $\ell(y, \hat{y})$ is a prior distribution and $Z(x, y)$ the partition function. The surrogate loss function is then the negative log-likelihood under the parameters $\boldsymbol{\theta}$

$$\bar{\ell}_{log}(\boldsymbol{\theta}, x, y) = \ln \frac{1}{p(\hat{y}|x, y; \boldsymbol{\theta})}.$$

In structured SVMs and CRFs a convex loss function and a convex regularization are minimized.

## 3  One parameter extension of CRFs and Structured SVMs

In CRFs one aims to minimize the regularized negative log-likelihood of the conditional distribution $p(\hat{y}|x, y; \boldsymbol{\theta})$ which decomposes into the log-partition and the linear term $\boldsymbol{\theta}^\top \Phi(x, y)$. Hence the problem of minimizing the regularized loss in (1) with the loss function $\bar{\ell}_{log}$ can be written as

$$\text{(CRF)} \qquad \min_{\boldsymbol{\theta}} \left\{ \sum_{(x,y) \in \mathcal{S}} \ln Z(x, y) - \mathbf{d}^\top \boldsymbol{\theta} + \frac{C}{p} \|\boldsymbol{\theta}\|_p^p \right\},$$

where $(x, y) \in \mathcal{S}$ ranges over training pairs and $\mathbf{d} = \sum_{(x,y) \in \mathcal{S}} \Phi(x, y)$ is the vector of empirical means.

Structured SVMs aim at minimizing the regularized hinge loss $\bar{\ell}_{hinge}(\boldsymbol{\theta}, x, y)$, which measures the loss of the label $y_{\boldsymbol{\theta}}(x)$ that most violates the training pair $(x, y) \in \mathcal{S}$ by more than $\ell(y, y_{\boldsymbol{\theta}}(x))$. Since $y_{\boldsymbol{\theta}}(x)$ is independent of the training label $y$, the structured SVM program takes the form:

(structured SVM)
$$\min_{\boldsymbol{\theta}} \left\{ \sum_{(x,y) \in \mathcal{S}} \max_{\hat{y} \in \mathcal{Y}} \left\{ \ell(y, \hat{y}) + \boldsymbol{\theta}^\top \Phi(x, \hat{y}) \right\} - \mathbf{d}^\top \boldsymbol{\theta} + \frac{C}{p} \|\boldsymbol{\theta}\|_p^p \right\},$$

where $(x, y) \in \mathcal{S}$ ranges over the training pairs, and $\mathbf{d}$ is the vector of empirical means.

In the following we deal with both structured prediction tasks (i.e., structured SVMs and CRFs) as two instances of the same framework, by extending the partition function to norms, namely $Z_\epsilon(x, y) = \| \exp \left( \ell(y, \hat{y}) + \boldsymbol{\theta}^\top \Phi(x, \hat{y}) \right) \|_{1/\epsilon}$, where the norm is computed for the vector ranging over $\hat{y} \in \mathcal{Y}$. Using the norm formulation we move from the partition function, for $\epsilon = 1$, to the maximum over the exponential function for $\epsilon = 0$. Equivalently, we relate the log-partition and the max-function by the soft-max function

$$\ln Z_\epsilon(x, y) = \epsilon \ln \sum_{\hat{y} \in \mathcal{Y}} \exp \left( \frac{\ell(y, \hat{y}) + \boldsymbol{\theta}^\top \Phi(x, \hat{y})}{\epsilon} \right) \tag{2}$$

For $\epsilon = 1$ the soft-max function reduces to the log-partition function, and for $\epsilon = 0$ it reduces to the max-function. Moreover, when $\epsilon \to 0$ the soft-max function is a smooth approximation of the max-function, in the same way the $\ell_{1/\epsilon}$-norm is a smooth approximation of the $\ell_\infty$-norm. This smooth approximation of the max-function is used in different areas of research [8]. We thus define the structured prediction problem as

(structured-prediction)
$$\min_{\boldsymbol{\theta}} \left\{ \sum_{(x,y) \in \mathcal{S}} \ln Z_\epsilon(x, y) - \mathbf{d}^\top \boldsymbol{\theta} + \frac{C}{p} \|\boldsymbol{\theta}\|_p^p \right\}, \tag{3}$$

which is a one-parameter extension of CRFs and structured SVMs, i.e., $\epsilon = 1$ and $\epsilon = 0$ respectively. Similarly to CRFs and structured SVMs [11, 16], one can use gradient methods to optimize structured prediction. The gradient of $\theta_r$ takes the form

$$\sum_{(x,y) \in \mathcal{S}} \sum_{\hat{y}} p_\epsilon(\hat{y}|x, y; \boldsymbol{\theta}) \phi_r(x, \hat{y}) - d_r + |\theta_r|^{p-1} \text{sign}(\theta_r), \tag{4}$$

where
$$p_\epsilon(\hat{y}|x, y; \boldsymbol{\theta}) = \frac{1}{Z_\epsilon(x, y)^{1/\epsilon}} \exp \left( \frac{\ell(y, \hat{y}) + \boldsymbol{\theta}^\top \Phi(x, \hat{y})}{\epsilon} \right) \tag{5}$$

is a probability distribution over the possible labels $\hat{y} \in \mathcal{Y}$. When $\epsilon \to 0$ this probability distribution gets concentrated around its maximal values, since all its elements are raised to the power of a very large number (i.e., $1/\epsilon$). Therefore for $\epsilon = 0$ we get a structured SVM subgradient.

In many real-life applications the labels $y \in \mathcal{Y}$ are $n$-tuples, $y = (y_1, ..., y_n)$, hence there are exponentially many labels in $\mathcal{Y}$. The feature maps usually describe relations between subsets of label elements $y_\alpha \subset \{y_1, ..., y_n\}$, and local interactions on single label elements $y_v$, namely

$$\phi_r(x, \hat{y}_1, ..., \hat{y}_n) = \sum_{v \in V_{r,x}} \phi_{r,v}(x, \hat{y}_v) + \sum_{\alpha \in E_{r,x}} \phi_{r,\alpha}(x, \hat{y}_\alpha). \tag{6}$$

Each feature $\phi_r(x, \hat{y})$ can be described by its *factor graph* $G_{r,x}$, a bipartite graph with one set of nodes corresponding to $V_{r,x}$ and the other set corresponds to $E_{r,x}$. An edge connects a single label node $v \in V_{r,x}$ with a subset of label nodes $\alpha \in E_{r,x}$ if and only if $y_v \in y_\alpha$. In the following we consider the factor graph $G = \cup_r G_r$ which is the union of all factor graphs. We denote by $N(v)$ and $N(\alpha)$ the set of neighbors of $v$ and $\alpha$ respectively, in the factor graph $G$. For clarity in the presentation we consider fully factorized loss $\ell(y, \hat{y}) = \sum_{v=1}^{n} \ell_v(y_v, \hat{y}_v)$, although our derivation naturally extends to any graphical model representing the interactions $\ell(y, \hat{y})$.

To compute the soft-max and the marginal probabilities, $p_\epsilon(\hat{y}_v|x, y; \boldsymbol{\theta})$ and $p_\epsilon(\hat{y}_\alpha|x, y; \boldsymbol{\theta})$, exponentially many labels have to be considered. This is in general computationally prohibitive, and thus one has to rely on inference and message-passing algorithms. When the factor graph has no cycles inference can be efficiently computed using belief propagation, but in the presence of cycles inference can only be approximated [25, 26, 7, 5, 13]. There are two main problems when dealing with graphs with cycles and approximate inference: efficiency and accuracy. For graphs with cycles there are no guarantees on the number of steps the message-passing algorithm requires till convergence, therefore it is computationally costly to run it as a subroutine. Moreover, as these message-passing algorithms have no guarantees on the quality of their solution, the gradient and the objective function can only be approximated, and one cannot know if the update rule decreased or increased the structured prediction objective. In contrast, in this work we propose to approximate the structured prediction problem and to efficiently solve the approximated problem exactly using message-passing. Intuitively, we suggest a principled way to run the approximate inference updates for few steps, while re-using the messages of previous steps to extract intermediate beliefs. These beliefs are used to update $\theta_r$, although the intermediate beliefs may not agree on their marginal probabilities. This allows us to efficiently learn graphical models with large number of parameters.

## 4 Approximate Structured Prediction

The structured prediction objective in (3) and its gradients defined in (4) cannot be computed efficiently for general graphs since both involve computing the soft-max function, $\ln Z_\epsilon(x, y)$, and the marginal probabilities, $p_\epsilon(\hat{y}_v|x, y; \boldsymbol{\theta})$ and $p_\epsilon(\hat{y}_\alpha|x, y; \boldsymbol{\theta})$, which take into account exponentially many elements $\hat{y} \in Y$. In the following we suggest an intuitive approximation for structured prediction, based on its dual formulation.

Since the dual of the soft-max is the entropy barrier, it follows that the dual program for structured prediction is governed by the entropy function of the probabilities $p_{x,y}(\hat{y})$. The following duality formulation is known for CRFs when $\epsilon = 1$ with $\ell_2^2$ regularization, and for structured SVM when $\epsilon = 0$ with $\ell_2^2$ regularization, [11, 21, 1]. Here we derive the dual program for every $\epsilon$ and every $\ell_p^p$ regularization using conjugate duality:

**Claim 1** *The dual program of the structured prediction program in (3) takes the form*

$$\max_{p_{x,y}(\hat{y}) \in \Delta_\mathcal{Y}} \sum_{(x,y) \in \mathcal{S}} \left( \epsilon H(\mathbf{p}_{x,y}) + \sum_{\hat{y}} p_{x,y}(\hat{y}) \ell(y, \hat{y}) \right) - \frac{C^{1-q}}{q} \left\| \sum_{(x,y) \in \mathcal{S}} \sum_{\hat{y} \in Y} p_{x,y}(\hat{y}) \Phi(x, \hat{y}) - \mathbf{d} \right\|_q^q,$$

*where $\Delta_\mathcal{Y}$ is the probability simplex over $\mathcal{Y}$ and $H(\mathbf{p}_{x,y}) = -\sum_{\hat{y}} p_{x,y}(\hat{y}) \ln p_{x,y}(\hat{y})$ is the entropy.*

**Proof:** In [6] □

When $\epsilon = 1$ the CRF dual program reduces to the well-known duality relation between the log-likelihood and the entropy. When $\epsilon = 0$ we obtain the dual formulation of structured SVM which emphasizes the duality relation between the max-function and the probability simplex. In general, Claim 1 describes the relation between the soft-max function and the entropy barrier over the probability simplex.

The dual program in Claim 1 considers the probabilities $p_{x,y}(\hat{y})$ over exponentially many labels $\hat{y} \in \mathcal{Y}$, as well as their entropies $H(\mathbf{p}_{x,y})$. However, when we take into account the graphical model imposed by the features, $G_{r,x}$, we observe that the linear terms in the dual formulation consider the marginals probabilities $p_{x,y}(\hat{y}_v)$ and $p_{x,y}(\hat{y}_\alpha)$. We thus propose to replace the marginal probabilities with their corresponding beliefs, and to replace the entropy term by the local entropies $\sum_\alpha c_\alpha H(\mathbf{b}_{x,y,\alpha}) + \sum_v c_v H(\mathbf{b}_{x,y,v})$ over the beliefs. Whenever $\epsilon, c_v, c_\alpha \geq 0$, the approximated dual is concave and it corresponds to a convex dual program. By deriving its dual we obtain our approximated structured prediction, for which we construct an efficient algorithm in Section 5.

| | Gaussian noise | | | | Bimodal noise | | | |
|---|---|---|---|---|---|---|---|---|
| | $I_1$ | $I_2$ | $I_3$ | $I_4$ | $I_1$ | $I_2$ | $I_3$ | $I_4$ |
| LBP-SGD | 2.7344 | 2.4707 | 3.2275 | 2.3193 | 5.2905 | 4.4751 | 6.8164 | 7.2510 |
| LBP-SMD | 2.7344 | 2.4731 | 3.2324 | 2.3145 | 5.2954 | 4.4678 | 6.7578 | 7.2583 |
| LBP-BFGS | 2.7417 | 2.4194 | 3.1299 | 2.4023 | 5.2148 | 4.3994 | 6.0278 | 6.6211 |
| MF-SGD | 3.0469 | 3.0762 | 4,1382 | 2.9053 | 10.0488 | 41.0718 | 29.6338 | 53.6035 |
| MF-SMD | 2.9688 | 3.0640 | 3.8721 | 14.4360 | – | – | – | – |
| MF-BFGS | 3.0005 | 2.7783 | 3.6157 | 2.4780 | 5.2661 | 4.6167 | 6.4624 | 7.2510 |
| Ours | **0.0488** | **0.0073** | **0.1294** | **0.1318** | **0.0537** | **0.0244** | **0.1221** | **0.9277** |

Figure 1: **Gaussian and bimodal noise**: Comparison of our approach to loopy belief propagation and mean field approximations when optimizing using BFGS, SGD and SMD. Note that our approach significantly outperforms all the baselines. MF-SMD did not work for Bimodal noise.

**Theorem 1** *The approximation of the structured prediction program in (3) takes the form*

$$
\min_{\lambda_{x,y,v\to\alpha},\boldsymbol{\theta}} \quad \sum_{(x,y)\in\mathcal{S},v} \epsilon c_v \ln \sum_{\hat{y}_v} \exp\left( \frac{\ell_v(y_v,\hat{y}_v) + \sum_{r:v\in V_{r,x}} \theta_r \phi_{r,v}(x,\hat{y}_v) - \sum_{\alpha\in N(v)} \lambda_{x,y,v\to\alpha}(\hat{y}_v)}{\epsilon c_v} \right)
$$

$$
+ \sum_{(x,y)\in\mathcal{S},\alpha} \epsilon c_\alpha \ln \sum_{\hat{y}_\alpha} \exp\left( \frac{\sum_{r:\alpha\in E_r} \theta_r \phi_{r,\alpha}(x,\hat{y}_\alpha) + \sum_{v\in N(\alpha)} \lambda_{x,y,v\to\alpha}(\hat{y}_v)}{\epsilon c_\alpha} \right) - \mathbf{d}^\top\boldsymbol{\theta} - \frac{C}{p}\|\boldsymbol{\theta}\|_p^p
$$

**Proof:** In [6] □

# 5 Message-Passing Algorithm for Approximated Structured Prediction

In the following we describe a block coordinate descent algorithm for the approximated structured prediction program of Theorem 1. Coordinate descent methods are appealing as they optimize a small number of variables while holding the rest fixed, therefore they are efficient and can be easily parallelized. Since the primal program is lower bounded by the dual program, the primal objective function is guaranteed to converge. We begin by describing how to find the optimal set of variables related to a node $v$ in the graphical model, namely $\lambda_{x,y,v\to\alpha}(\hat{y}_v)$ for every $\alpha\in N(v)$, every $\hat{y}_v$ and every $(x,y)\in\mathcal{S}$.

**Lemma 1** *Given a vertex $v$ in the graphical model, the optimal $\lambda_{x,y,v\to\alpha}(\hat{y}_v)$ for every $\alpha\in N(v), \hat{y}_v\in\mathcal{Y}_v, (x,y)\in\mathcal{S}$ in the approximated program of Theorem 1 satisfies*

$$
\mu_{x,y,\alpha\to v}(\hat{y}_v) = \epsilon c_\alpha \ln\left( \sum_{\hat{y}_\alpha\setminus\hat{y}_v} \exp\left( \frac{\sum_{r:\alpha\in E_{r,x}} \theta_r \phi_{r,\alpha}(x,\hat{y}_\alpha) + \sum_{u\in N(\alpha)\setminus v} \lambda_{x,y,u\to\alpha}(\hat{y}_u)}{\epsilon c_\alpha} \right) \right)
$$

$$
\lambda_{x,y,v\to\alpha}(\hat{y}_v) = \frac{c_\alpha}{\hat{c}_v}\left( \ell_v(y_v,\hat{y}_v) + \sum_{r:v\in V_{r,x}} \theta_r \phi_{r,v}(x,\hat{y}_v) + \sum_{\beta\in N(v)} \mu_{x,y,\beta\to v}(\hat{y}_v) \right) - \mu_{x,y,\alpha\to v}(\hat{y}_v) + c_{x,y,v\to\alpha}
$$

*for every constant $c_{x,y,v\to\alpha}$[1], where $\hat{c}_v = c_v + \sum_{\alpha\in N(v)} c_\alpha$. In particular, if either $\epsilon$ and/or $c_\alpha$ are zero then $\mu_{x,y,\alpha\to v}$ corresponds to the $\ell_\infty$ norm and can be computed by the max-function. Moreover, if either $\epsilon$ and/or $c_\alpha$ are zero in the objective, then the optimal $\lambda_{x,y,v\to\alpha}$ can be computed for any arbitrary $c_\alpha > 0$, and similarly for $c_v > 0$.*

**Proof:** In [6] □

It is computationally appealing to find the optimal $\lambda_{x,y,v\to\alpha}(\hat{y}_v)$. When the optimal value cannot be found, one usually takes a step in the direction of the negative gradient and the objective function needs to be computed to ensure that the chosen step size reduces the objective. Obviously, computing the objective function at every iteration significantly slows the algorithm. When the optimal $\lambda_{x,y,v\to\alpha}(\hat{y}_v)$ can be found, the block coordinate descent algorithm can be executed efficiently in distributed manner, since every $\lambda_{x,y,v\to\alpha}(\hat{y}_v)$ can be computed independently. The only interactions occur when computing the normalization step $c_{x,y,v\to\alpha}$. This allows for easy computation in GPUs.

We now turn to describe how to change $\boldsymbol{\theta}$ in order to improve the approximated structured prediction. Since we cannot find the optimal $\theta_r$ while holding the rest fixed, we perform a step in the direction

of the negative gradient, when $\epsilon, c_\alpha, c_i$ are positive, or in the direction of the subgradient otherwise. We choose the step size $\eta$ to guarantee a descent on the objective.

**Lemma 2** *The gradient of the approximated structured prediction program in Theorem 1 with respect to $\theta_r$ equals to*

$$\sum_{(x,y)\in\mathcal{S},v\in V_{r,x},\hat{y}_v} b_{x,y,v}(\hat{y}_v)\phi_{r,v}(x,\hat{y}_v) + \sum_{(x,y)\in\mathcal{S},\alpha\in E_{r,x},\hat{y}_\alpha} b_{x,y,\alpha}(\hat{y}_\alpha)\phi_{r,\alpha}(x,\hat{y}_\alpha) - d_r + C\cdot|\theta_r|^{p-1}\cdot sign(\theta_r),$$

*where*

$$b_{x,y,v}(\hat{y}_v) \propto \exp\left(\frac{\ell_v(y_v,\hat{y}_v) + \sum_{r:v\in V_{r,x}}\theta_r\phi_{r,v}(x,\hat{y}_v) - \sum_{\alpha\in N(v)}\lambda_{x,y,v\to\alpha}(\hat{y}_v)}{\epsilon c_v}\right)$$

$$b_{x,y,\alpha}(\hat{y}_\alpha) \propto \exp\left(\frac{\sum_{r:\alpha\in E_{r,x}}\theta_r\phi_{r,\alpha}(x,\hat{y}_\alpha) + \sum_{v\in N(\alpha)}\lambda_{x,y,v\to\alpha}(\hat{y}_v)}{\epsilon c_\alpha}\right)$$

*However, if either $\epsilon$ and/or $c_\alpha$ equal zero, then the beliefs $b_{x,y,\alpha}(\hat{y}_\alpha)$ can be taken from the set of probability distributions over support of the max-beliefs, namely $b_{x,y,\alpha}(\hat{y}_\alpha^*) > 0$ only if $\hat{y}_\alpha^* \in argmax_{\hat{y}_\alpha}\left\{\sum_{r:\alpha\in E_{r,x}}\theta_r\phi_{r,\alpha}(x,\hat{y}_\alpha) + \sum_{v\in N(\alpha)}\lambda_{x,y,v\to\alpha}(\hat{y}_\alpha)\right\}$. Similarly for $b_{x,y,v}(\hat{y}_v^*)$ whenever $\epsilon$ and/or $c_v$ equal zero.*

**Proof:** In [6] □

Lemmas 1 and 2 describe the coordinate descent algorithm for the approximated structured prediction in Theorem 1. We refer the reader to [6] for a summary of our algorithm.

The coordinate descent algorithm is guaranteed to converge, as it monotonically decreases the approximated structured prediction objective in Theorem 1, which is lower bounded by its dual program. However, convergence to the global minimum cannot be guaranteed in all cases. In particular, for $\epsilon = 0$ the coordinate descent on the approximated structured SVMs is not guaranteed to converge to its global minimum, unless one uses subgradient methods which are not monotonically decreasing. Moreover, even when we are guaranteed to converge to the global minimum, i.e., $\epsilon, c_\alpha, c_v > 0$, the sequence of variables $\lambda_{x,y,v\to\alpha}(\hat{y}_v)$ generated by the algorithm is not guaranteed to converge to an optimal solution, nor to be bounded. As a trivial example, adding an arbitrary constant to the variables, $\lambda_{x,y,v\to\alpha}(\hat{y}_v) + c$, does not change the objective value, hence the algorithm can generate non-decreasing unbounded sequences. However, the beliefs generated by the algorithm are bounded and guaranteed to converge to the solution of the dual approximated structured prediction problem.

**Claim 2** *The block coordinate descent algorithm in lemmas 1 and 2 monotonically reduces the approximated structured prediction objective in Theorem 1, therefore the value of its objective is guaranteed to converge. Moreover, if $\epsilon, c_\alpha, c_v > 0$, the objective is guaranteed to converge to the global minimum, and its sequence of beliefs are guaranteed to converge to the unique solution of the approximated structured prediction dual.*

**Proof:** In [6] □

The convergence result has a practical implication, describing the ways we can estimate the convergence of the algorithm, either by the primal objective, the dual objective or the beliefs. The approximated structured prediction can also be used for non-concave entropy approximations, such as the Bethe entropy, where $c_\alpha > 0$ and $c_v < 0$. In this case the algorithm is well defined, and its stationary points correspond to the stationary points of the approximated structured prediction and its dual. Intuitively, this statement holds since the coordinate descent algorithm iterates over points $\lambda_{x,y,v\to\alpha}(\hat{y}_v), \theta_r$ with vanishing gradients. Equivalently the algorithm iterates over saddle points $\lambda_{x,y,v\to\alpha}(\hat{y}_v), b_{x,y,v}(\hat{y}_v), b_{x,y,\alpha}(\hat{y}_\alpha)$ and $(\theta_r, z_r)$ of the Lagrangian defined in Theorem 1. Whenever the dual program is concave these saddle points are optimal points of the convex primal, but for non-concave dual the algorithm iterates over saddle points. This is summarized in the claim below:

**Claim 3** *Whenever the approximated structured prediction is non convex, i.e., $\epsilon, c_\alpha > 0$ and $c_v < 0$, the algorithm in lemmas 1 and 2 is not guaranteed to converge, but whenever it converges it reaches a stationary point of the primal and dual approximated structured prediction programs.*

**Proof:** In [6] □

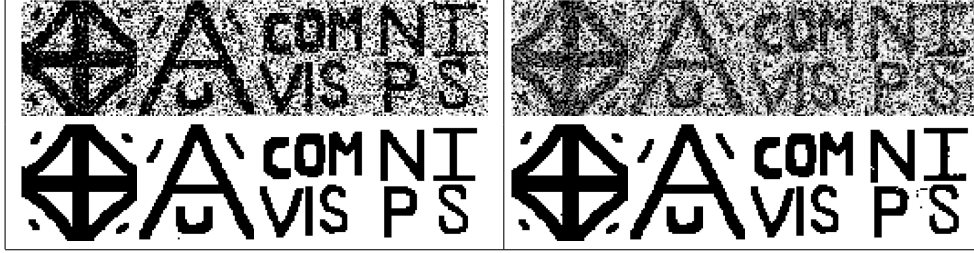

Figure 2: **Denoising results**: Gaussian (left) and Bimodal (right) noise.

# 6   Experimental evaluation

We performed experiments on 2D grids since they are widely used to represent images, and have many cycles. We first investigate the role of $\epsilon$ in the accuracy and running time of our algorithm, for fixed $c_\alpha, c_v = 1$. We used a $10 \times 10$ binary image and randomly generated 10 corrupted samples flipping every bit with 0.2 probability. We trained the model using CRF, structured-SVM and our approach for $\epsilon = \{1, 0.5, 0.01, 0\}$, ranging from approximated CRFs ($\epsilon = 1$) to approximated structured SVM ($\epsilon = 0$) and its smooth version ($\epsilon = 0.01$). The runtime for CRF and structured-SVM is order of magnitudes larger than our method since they require exact inference for every training example and every iteration of the algorithm. For the approximated structured prediction, the runtimes are $323, 324, 326, 294$ seconds for $\epsilon = \{1, 0.5, 0.01, 0\}$ respectively. As $\epsilon$ gets smaller the runtime slightly increases, but it decreases for $\epsilon = 0$ since the $\ell_\infty$ norm is computed efficiently using the max function. However, $\epsilon = 0$ is less accurate than $\epsilon = 0.01$; When the approximated structured SVM converges, the gap between the primal and dual objectives was 1.3, and only $10^{-5}$ for $\epsilon > 0$. This is to be expected since the approximated structured SVM is non-smooth (Claim 2), and we did not used subgradient methods to ensure convergence to the optimal solution.

We generated test images in a similar fashion while using the same $\epsilon$ for training and testing. In this setting both CRF and structured-SVM performed well, with 2 misclassifications. For the approximated structured prediction, we obtained 2 misclassifications for $\epsilon > 0$. We also evaluated the quality of the solution using different values of $\epsilon$ for training and inference [24]. When predicting with smaller $\epsilon$ than the one used for learning the results are marginally worse than when predicting with the same $\epsilon$. However, when predicting with larger $\epsilon$, the results get significantly worse, e.g., learning with $\epsilon = 0.01$ and predicting with $\epsilon = 1$ results in 10 errors, and only 2 when $\epsilon = 0.01$.

The main advantage of our algorithm is that it can efficiently learn many parameters. We now compared in a $5 \times 5$ dataset a model learned with different parameters for every edge and vertex ($\approx 300$ parameters) and a model learned with parameters shared among the vertices and edges (2 parameters for edges and 2 for vertices) [9]. Using large number of parameters increases performance: sharing parameters resulted in 16 misclassifications, while optimizing over the 300 parameters resulted in 2 errors. Our algorithm avoids overfitting in this case, we conjecture it is due to the regularization.

We now compare our approach to state-of-the-art CRF solvers on the binary image dataset of [9] that consists of 4 different $64 \times 64$ base images. Each base image was corrupted 50 times with each type of noise. Following [23], we trained different models to denoise each individual image, using 40 examples for training and 10 for test. We compare our approach to approximating the conditional likelihood using loopy belief propagation (LBP) and mean field approximation (MF). For each of these approximations, we use stochastic gradient descent (SGD), stochastic meta-descent (SMD) and BFGS to learn the parameters. We do not report pseudolikelihood (PL) results since it did not work. The same behavior of PL was noticed by [23]. To reduce the computational complexity and the chances of convergence, [9, 23] forced their parameters to be shared across all nodes such that $\forall i, \theta_i = \theta^n$ and $\forall i, \forall j \in N(i), \; \theta_{ij} = \theta^e$. In contrast, since our approach is efficient, we can exploit the full flexibility of the graph and learn more than $10,000$ parameters. This is computationally prohibitive with the baselines. We use the pixel values as node potentials and an Ising model with only bias for the edge potentials, i.e., $\phi_{i,j} = [1, -1; -1, 1]$. For all experiments we use $\epsilon = 1$, and $p = 2$. For the baselines, we use the code, features and optimal parameters of [23].

Under the first noise model, each pixel was corrupted via i.i.d. Gaussian noise with mean 0 and standard deviation of 0.3. Fig. 1 depicts test error in (%) for the different base images (i.e., $I_1, \ldots, I_4$). Note that our approach outperforms considerably the loopy belief propagation and mean field approximations for all optimization criteria (BFGS, SGD, SMD). For example, for the first base image the error of our approach is $0.0488\%$, which is equivalent to a 2 pixels error on average. In contrast

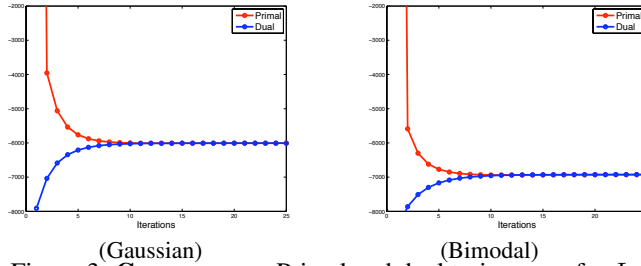

(Gaussian)                    (Bimodal)

Figure 3: **Convergence**. Primal and dual train errors for $I_1$.

the best baseline gets 112 pixels wrong on average. Fig. 2 (left) depicts test examples as well as our denoising results. Note that our approach is able to cope with large amounts of noise.

Under the second noise model, each pixel was corrupted with an independent mixture of Gaussians. For each class, a mixture of 2 Gaussians with equal mixing weights was used, yielding the Bimodal noise. The mixture model parameters were $(0.08, 0.03)$ and $(0.46, 0.03)$ for the first class and $(0.55, 0.02)$ and $(0.42, 0.10)$ for the second class, with $(a, b)$ a Gaussian with mean $a$ and standard deviation $b$. Fig. 1 depicts test error in $(\%)$ for the different base images. As before, our approach outperforms all the baselines. We do not report MF-SMD results since it did not work. Denoised images are shown in Fig. 2 (right). We now show how our algorithm converges in a few iterations. Fig. 3 depicts the primal and dual training errors as a function of the number of iterations. Note that our algorithm converges, and the dual and primal values are very tight after a few iterations.

## 7 Related Work

For the special case of CRFs, the idea of approximating the entropy function with local entropies appears in [24, 3]. In particular, [24] proved that using a concave entropy approximation gives robust prediction. [3] optimized the non-concave Bethe entropy $c_\alpha = 1, c_v = 1 - |N(v)|$, by repeatedly maximizing its concave approximation, thus converging in few concave iterations. Our work differs from these works in two aspects: we derive an efficient algorithm in Section 5 for the concave approximated program ($c_\alpha, c_v > 0$) and our framework and algorithm include structured SVMs, as well as their smooth approximation when $\epsilon \to 0$.

Some forms of approximated structured prediction were investigated for the special cases of CRFs. In [18] a similar program was used, but without the Lagrange multipliers $\lambda_{x,y,v\to\alpha}(\hat{y}_v)$ and no regularization, i.e., $C = 0$. As a result the local log-partition functions are unrelated, and efficient counting algorithm can be used for learning. In [3] a different approximated program was derived for $c_\alpha = 1, c_v = 0$ which was solved by the BFGS convex solver. Our work is different as it considers efficient algorithms for approximated structured prediction which take advantage of the graphical model by sending messages along its edges. We show in the experiments that this significantly improves the run-time of the algorithm. Also, our approximated structured prediction includes as special cases approximated CRF, for $\epsilon = 1$, and approximated structured SVM, for $\epsilon = 0$. Moreover, we describe how to smoothly approximate the structured SVMs to avoid the shortcomings of subgradient methods, by simply setting $\epsilon \to 0$.

Some forms of approximated structured SVMs were dealt in [19] with the structured SMO algorithm. Independently, [14] presented an approximated structured SVMs program and a message passing algorithm, which reduce to Theorem 1 and Lemma 1 with $\epsilon = 0$ and $c_\alpha = 1, c_v = 1$. However, in this algorithm the messages are not guaranteed to be bounded. They main difference of [14] from our work is that they lack the dual formulation, which we use to prove that the structured SVM smooth approximation, with $\epsilon \to 0$, is guaranteed to converge to optimum and that the dual variables, i.e. the beliefs, are guaranteed to converge to the optimal beliefs. The relation between the margin and the soft-max is similar to the one used in [17]. Independently, [4, 15] described the connection between structured SVMs loss and CRFs loss. [15] also presented the one-parameter extension of CRFs and structured SVMs described in (3).

## 8 Conclusion and Discussion

In this paper we have related CRFs and structured SVMs and shown that the soft-max, a variant of the log-partition function, approximates smoothly the structured SVM hinge loss. We have also proposed an approximation for structured prediction problems based on local entropy approximations and derived an efficient message-passing algorithm that is guaranteed to converge, even for general graphs. We have demonstrated the effectiveness of our approach to learn graphs with large number of parameters. We plan to investigate other domains of application such as image segmentation.

## Footnotes

[1]For numerical stability in our algorithm we set $c_{x,y,v\to\alpha}$ such that $\sum_{\hat{y}_v} \lambda_{x,y,v\to\alpha}(\hat{y}_v) = 0$

# References

[1] M. Collins, A. Globerson, T. Koo, X. Carreras, and P.L. Bartlett. Exponentiated gradient algorithms for conditional random fields and max-margin markov networks. *JMLR*, 9:1775–1822, 2008.

[2] T. Finley and T. Joachims. Training structural SVMs when exact inference is intractable. In *ICML*, pages 304–311. ACM, 2008.

[3] V. Ganapathi, D. Vickrey, J. Duchi, and D. Koller. Constrained approximate maximum entropy learning of markov random fields. In *UAI*, 2008.

[4] K. Gimpel and N.A. Smith. Softmax-margin CRFs: Training log-linear models with cost functions. In *Human Language Technologies: The 2010 Annual Conference of the North American Chapter of the Association for Computational Linguistics*, pages 733–736. Association for Computational Linguistics, 2010.

[5] T. Hazan and A. Shashua. Norm-Product Belief Propagation: Primal-Dual Message-Passing for Approximate Inference. *Arxiv preprint arXiv:0903.3127*, 2009.

[6] T. Hazan and R. Urtasun. Approximated Structured Prediction for Learning Large Scale Graphical Models. *Arxiv preprint arXiv:1006.2899*, 2010.

[7] T. Heskes. Convexity arguments for efficient minimization of the Bethe and Kikuchi free energies. *Journal of Artificial Intelligence Research*, 26(1):153–190, 2006.

[8] J.K. Johnson, D.M. Malioutov, and A.S. Willsky. Lagrangian relaxation for MAP estimation in graphical models. In *Proceedings of the Allerton Conference on Control, Communication and Computing*. Citeseer, 2007.

[9] S. Kumar and M. Hebert. Discriminative Fields for Modeling Spatial Dependencies in Natural Images. In *Neural Information Processing Systems*. MIT Press, Cambridge, MA, 2003.

[10] J. Lafferty, A. McCallum, and F. Pereira. Conditional random fields: Probabilistic models for segmenting and labeling sequence data. In *ICML*, pages 282–289, 2001.

[11] G. Lebanon and J. Lafferty. Boosting and maximum likelihood for exponential models. *NIPS*, 1:447–454, 2002.

[12] A. Levin and Y. Weiss. Learning to Combine Bottom-Up and Top-Down Segmentation. In *European Conference on Computer Vision*, 2006.

[13] T. Meltzer, A. Globerson, and Y. Weiss. Convergent message passing algorithms-a unifying view. In *UAI*, 2009.

[14] O. Meshi, D. Sontag, T. Jaakkola, and A. Globerson. Learning Efficiently with Approximate Inference via Dual Losses. In *Proc. ICML*. Citeseer, 2010.

[15] P. Pletscher, C. Ong, and J. Buhmann. Entropy and Margin Maximization for Structured Output Learning. *Machine Learning and Knowledge Discovery in Databases*, pages 83–98, 2010.

[16] N. Ratliff, J.A. Bagnell, and M. Zinkevich. Subgradient methods for maximum margin structured learning. In *ICML Workshop on Learning in Structured Output Spaces*, 2006.

[17] F. Sha and L.K. Saul. Large margin hidden Markov models for automatic speech recognition. *Advances in neural information processing systems*, 19:1249, 2007.

[18] C. Sutton and A. McCallum. Piecewise training for structured prediction. *Machine Learning*, 77(2):165–194, 2009.

[19] B. Taskar. *Learning structured prediction models: a large margin approach*. PhD thesis, Stanford, CA, USA, 2005. Adviser-Koller, Daphne.

[20] B. Taskar, P. Abbeel, and D. Koller. Discriminative probabilistic models for relational data. In *UAI*, pages 895–902. Citeseer, 2002.

[21] B. Taskar, C. Guestrin, and D. Koller. Max-margin Markov networks. *NIPS*, 16:51, 2004.

[22] B. Taskar, S. Lacoste-Julien, and M. I. Jordan. Structured prediction, dual extragradient and Bregman projections. *JMLR*, 7:1653–1684, 2006.

[23] S. Vishwanathan, N. Schraudolph, M. Schmidt, and K. Murphy. Accelerated Training of Conditional Random Fields with Stochastic Meta-Descent . In *International Conference in Machine Learning*, 2006.

[24] M.J. Wainwright. Estimating the Wrong Graphical Model: Benefits in the Computation-Limited Setting. *JMLR*, 7:1859, 2006.

[25] M.J. Wainwright and M.I. Jordan. Graphical models, exponential families, and variational inference. *Foundations and Trends® in Machine Learning*, 1(1-2):1–305, 2008.

[26] J.S. Yedidia, W. T. Freeman, and Y. Weiss. Constructing free-energy approximations and generalized belief propagation algorithms. *Transactions on Information Theory*, 51(7):2282–2312, 2005.

